# A Neural Oscillator Model of Auditory Selective Attention

**Stuart N. Wrigley and Guy J. Brown**

Department of Computer Science, University of Sheffield, Regent Court,
211 Portobello Street, Sheffield S1 4DP, UK.

s.wrigley@dcs.shef.ac.uk, g.brown@dcs.shef.ac.uk

## Abstract

A model of auditory grouping is described in which auditory attention plays a key role. The model is based upon an oscillatory correlation framework, in which neural oscillators representing a single perceptual stream are synchronised, and are desynchronised from oscillators representing other streams. The model suggests a mechanism by which attention can be directed to the high or low tones in a repeating sequence of tones with alternating frequencies. In addition, it simulates the perceptual segregation of a mistuned harmonic from a complex tone.

## 1  Introduction

In virtually all listening situations, we are exposed to a mixture of sound energy from multiple sources. Hence, the auditory system must separate an acoustic mixture in order to create a perceptual description of each sound source. It has been proposed that this process of *auditory scene analysis* (ASA) [2] takes place in two conceptual stages: *segmentation* in which the acoustic mixture is separated into its constituent 'atomic' units, followed by *grouping* in which units that are likely to have arisen from the same source are recombined. The perceptual 'object' produced by auditory grouping is called a *stream*. Each stream describes a single sound source.

Few studies have investigated the role of attention in ASA; typically, ASA is seen as a precursor to attentional mechanisms, which simply select one stream as the attentional focus. Recently, however, it has been suggested that attention plays a much more prominent role in ASA. Carlyon *et al.* [4] investigated how attention influences auditory grouping with the use of a rapidly repeating sequence of high and low tones. It is known that high frequency separations and/or high presentation rates encourage the high tones and low tones to form separate streams, a phenomenon known as *auditory streaming* [2]. Carlyon *et al.* demonstrated that auditory streaming did not occur when listeners attended to an alternative stimulus presented simultaneously. However, when they were instructed to attend to the tone sequence, auditory streaming occurred as normal. From this, it was concluded that attention is required for stream formation and not only for stream selection.

It has been proposed that attention can be divided into two different levels [9]: low-level *exogenous* attention which groups acoustic elements to form streams, and a higher-level *endogenous* mechanism which performs stream selection. Exogenous attention may over-rule conscious (endogenous) selection (e.g. in response to a sudden loud bang). The work presented here incorporates these two types of attention into a model of auditory grouping (Figure 1). The model is based upon the oscillatory correlation theory [10], which suggests that neural oscillations encode auditory grouping. Oscillators corresponding to grouped auditory elements are synchronised, and are desynchronised from oscillators encoding other groups. This theory is supported by neurobiological findings that report

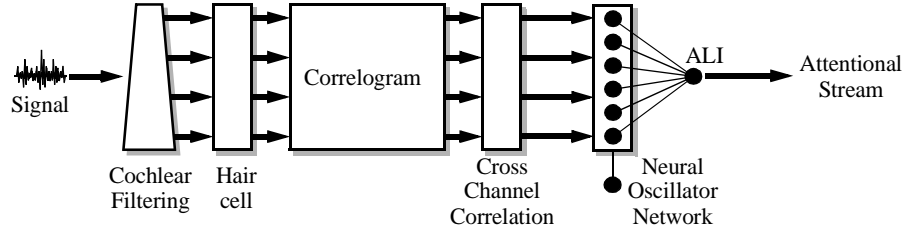

Figure 1: Schematic diagram of the model (the attentional leaky integrator is labelled ALI).

synchronised oscillations in the auditory system [6]. Within the oscillatory correlation framework, attentional selection can be implemented by synchronising attentional activity with the stream of interest.

## 2 The model

### 2.1 Auditory periphery

Cochlear filtering is modelled by a bank of 128 gammatone filters with centre frequencies equally spaced on the equivalent rectangular bandwidth (ERB) scale between 50 Hz and 2.5 kHz [3]. Auditory nerve firing rate is approximated by half-wave rectifying and square root compressing the output of each filter. Input to the model is sampled at a rate of 8 kHz.

### 2.2 Pitch and harmonicity analysis

It is known that a difference in fundamental frequency (F0) can assist the perceptual segregation of complex sounds [2]. Accordingly, the second stage of the model extracts pitch information from the simulated auditory nerve responses. This is achieved by computing the autocorrelation of the activity in each channel to form a correlogram [3]. At time $t$, the autocorrelation of channel $i$ with lag $\tau$ is given by:

$$A(i, t, \tau) = \sum_{k=0}^{P-1} r(i, t-k)r(i, t-k-\tau)w(k) \tag{1}$$

Here, $r$ is the auditory nerve activity. The autocorrelation for channel $i$ is computed using a 25 ms rectangular window $w$ ($P = 200$) with lag steps equal to the sampling period, up to a maximum lag of 20 ms. When the correlogram is summed across frequency, the resulting 'summary correlogram' exhibits a large peak at the lag corresponding to the fundamental period of the stimulus. An accurate estimate of the F0 is found by fitting a parabolic curve to the three samples centred on the summary peak.

The correlogram may also be used to identify formant and harmonic regions due to their similar patterns of periodicity [11]. This is achieved by computing the correlations between adjacent channels of the correlogram as follows:

$$C(i) = \frac{1}{L} \sum_{\tau=0}^{L-1} \hat{A}(i, t, \tau)\hat{A}(i+1, t, \tau) \tag{2}$$

Here, $\hat{A}(i, t, \tau)$ is the autocorrelation function of (1) which has been normalised to have zero mean and unity variance; $L$ is the maximum autocorrelation lag in samples ($L = 160$).

### 2.3 Neural oscillator network

The network consists of 128 oscillators and is based upon the two-dimensional locally excitatory globally inhibitory oscillator network (LEGION) of Wang [10], [11]. Within

LEGION, oscillators are synchronised by placing local excitatory links between them. Additionally, a global inhibitor receives excitation from each oscillator, and inhibits every oscillator in the network. This ensures that only one block of synchronised oscillators can be active at any one time. Hence, separate blocks of synchronised oscillators - which correspond to the notion of a segment in ASA - arise through the action of local excitation and global inhibition.

The model described here differs from Wang's approach [10] in three respects. Firstly, the network is one-dimensional rather than two-dimensional; we argue that this is more plausible. Secondly, excitatory links can be global as well as local; this allows harmonically-related segments to be grouped. Finally, we introduce an attentional leaky integrator (ALI), which selects one block of oscillators to become the attentional stream (i.e., the stream which is in the attentional foreground).

The building block of the network is a single oscillator, which consists of a reciprocally connected excitatory unit and inhibitory unit whose activities are represented by $x$ and $y$ respectively:

$$\dot{x} = 3x - x^3 + 2 - y + I_o \tag{3a}$$

$$\dot{y} = \varepsilon\left[\gamma\left(1 + \tanh\frac{x}{\beta}\right) - y\right] \tag{3b}$$

Here, $\varepsilon$, $\gamma$ and $\beta$ are parameters. Oscillations are stimulus dependent; they are only observed when $I_o > 0$, which corresponds to a periodic solution to (3) in which the oscillator cycles between an 'active' phase and a 'silent' phase. The system may be regarded as a model for the behaviour of a single neuron, or as a mean field approximation to a group of connected neurons. The input $I_o$ to oscillator $i$ is a combination of three factors: external input $I_r$, network activity and global inhibition as follows:

$$I_o = I_r - W_z S(z, \theta_z) + \sum_{k \neq i} W_{ik} S(x_k, \theta_x) \tag{4}$$

Here, $W_{ik}$ is the connection strength between oscillators $i$ and $k$; $x_k$ is the activity of oscillator $k$. The parameter $\theta_x$ is a threshold above which an oscillator can affect others in the network and $W_z$ is the weight of inhibition from the global inhibitor $z$. Similar to $\theta_x$, $\theta_z$ acts a threshold above which the global inhibitor can affect an oscillator. $S$ is a squashing function which compresses oscillator activity to be within a certain range:

$$S(n, \theta) = \frac{1}{1 + e^{-K(n - \theta)}} \tag{5}$$

Here, $K$ determines the sharpness of the sigmoidal function. The activity of the global inhibitor is defined as

$$\dot{z} = H\left(\sum_k S(x_k, \theta_x) - 0.1\right) - z \tag{6}$$

where $H$ is the Heaviside function ($H(n) = 1$ for $n \geq 0$, zero otherwise).

### 2.3.1 Segmentation

A block of channels are deemed to constitute a segment if the cross-channel correlation (2) is greater than 0.3 for every channel in the block. Cross-correlations are weighted by the energy of each channel in order to increase the contrast between spectral peaks and spectral dips. These segments are encoded by a binary mask, which is unity when a channel contributes to a segment and zero otherwise. To improve the resolution and separation of adjacent segments,

the cross-frequency spread of a segment is restricted to 3 channels. Oscillators within a segment are synchronized by excitatory connections. The external input ($I_r$) of an oscillator whose channel is a member of a segment is set to $I_{high}$ otherwise it is set to $I_{low}$.

### 2.3.2 Harmonicity grouping

Excitatory connections are made between segments if they are consistent with the current F0 estimate. A segment is classed as consistent with the F0 if a majority of its corresponding correlogram channels exhibit a significant peak at the fundamental period (ratio of peak height to channel energy greater than 0.46). A single connection is made between the centres of harmonically related segments subject to *old-plus-new* constraints.

The old-plus-new heuristic [2] refers to the auditory system's preference to '*interpret any part of a current group of acoustic components as a continuation of a sound that just occurred*'. This is incorporated into the model by attaching 'age trackers' to each channel of the network. Excitatory links are placed between harmonically related segments only if the two segments are of similar age. The age trackers are leaky integrators:

$$\dot{B}_k = d(g[M_k - B_k]^+ - [1 - H(M_k - B_k)]cB_k) \tag{7}$$

Here, $[n]^+ = n$ if $n \geq 0$ and $[n]^+ = 0$ otherwise. $M_k$ is the (binary) value of the segment mask at channel $k$; small values of $c$ and $d$ result in a slow rise ($d$) and slow decay ($c$) for the integrator. $g$ is a gain factor.

Consider two segments that start at the same time; the age trackers for their constituent channels receive the same input, so the values of $B_k$ will be the same. However, if two segments start at different times, the age trackers for the earlier segment will have already increased to a non-zero value when the second segment starts. This 'age difference' will dissipate over time, as the values of both sets of leaky integrators approach unity.

### 2.3.3 Attentional leaky integrator (ALI)

Each oscillator is connected to the attentional leaky integrator (ALI) by excitatory links; the strength of these connections is modulated by endogenous attention. Input to the ALI is given by:

$$\dot{ali} = H\left(\sum_k S(x_k, \theta_x)T_k - \theta_{ALI}\right) - ali \tag{8}$$

$\theta_{ALI}$ is a threshold above which network activity can influence the ALI. $T_k$ is an attentional weighting which is related to the endogenous interest at frequency $k$:

$$T_k = 1 - (1 - A_k)L \tag{9}$$

Here, $A_k$ is the endogenous interest at frequency $k$ and $L$ is the leaky integrator defined as:

$$\dot{L} = a(b[R - L]^+ - [1 - H(R - L)]fL) \tag{10}$$

Small values of $f$ and $a$ result in a slow rise ($a$) and slow decay ($f$) for the integrator. $b$ is a gain factor. $R = H(x_{max})$ where $x_{max}$ is the largest output activity of the network. The build-up of attentional interest is therefore stimulus dependent. The attentional interest itself is modelled as a Gaussian according to the gradient model of attention [7]:

$$A_k = max_{A_k} e^{-\frac{k-p}{2\sigma^2}} \tag{11}$$

Here, $A_k$ is the normalised attentional interest at frequency channel $k$ and $max_{Ak}$ is the maximum value that $A_k$ can attain. $p$ is the channel at which the peak of attentional interest occurs, and $\sigma$ determines the width of the peak.

A segment or group of segments are said to be attended to if their oscillatory activity coincides temporally with a peak in the ALI activity. Initially, the connection weights between the oscillator array and the ALI are strong: all segments feed excitation to the ALI, so all segments are attended to. During sustained activity, these weights relax toward the $A_k$ interest vector such that strong weights exist for channels of high attentional interest and low weights exist for channels of low attentional interest. ALI activity will only coincide with activity of the channels within the attentional interest peak and any harmonically related (synchronised) activity outside the $A_k$ peak. All other activity will occur within a trough of ALI activity. This behaviour allows both individual tones and harmonic complexes to be attended to using only a single $A_k$ peak.

The parameters for all simulations reported here were $\varepsilon = 0.4$, $\gamma = 6.0$, $\beta = 0.1$, $W_z = 0.5$, $\theta_z = 0.1$, $\theta_x = -0.5$ and $K = 50$, $d = 0.001$, $c = 5$, $g = 3$, $a = 0.0005$, $f = 5$, $b = 3$, $max_{Ak} = 1$, $\sigma = 3$, $\theta_{ALI} = 1.5$, $I_{low} = -5.0$, $I_{high} = 0.2$. The inter- and intra- segment connections have equal weights of 1.1.

## 3 Evaluation

Throughout this section, output from the model is represented by a 'pseudospectrogram' with time on the abscissa and frequency channel on the ordinate. Three types of information are superimposed on each plot. A gray pixel indicates the presence of a segment at a particular frequency channel, which is also equivalent to the external input to the corresponding oscillator: gray signifies $I_{high}$ (causing the oscillator to be stimulated) and white signifies $I_{low}$ (causing the oscillator to be unstimulated). Black pixels represent active oscillators (i.e. oscillators whose $x$ value exceeds a threshold value). At the top of each figure, ALI activity is shown. Any oscillators which are temporally synchronised with the ALI are considered to be in the attentional foreground.

### 3.1 Segregation of a component from a harmonic complex

Darwin *et al.* [5] investigated the effect of a mistuned harmonic upon the pitch of a 12 component complex tone. As the degree of mistuning of the fourth harmonic increased towards 4%, the shift in the perceived pitch of the complex also increased. This effect was less pronounced for mistunings of more than 4%; beyond 8% mistuning, little pitch shift was observed. Apparently, the pitch of a complex tone is calculated using only those channels which belong to the corresponding stream. When the harmonic is subject to mistunings below 8%, it is grouped with the rest of the complex and so can affect the pitch percept. Mistunings of greater than 8% cause the harmonic to be segregated into a second stream, and so it is excluded from the pitch percept.

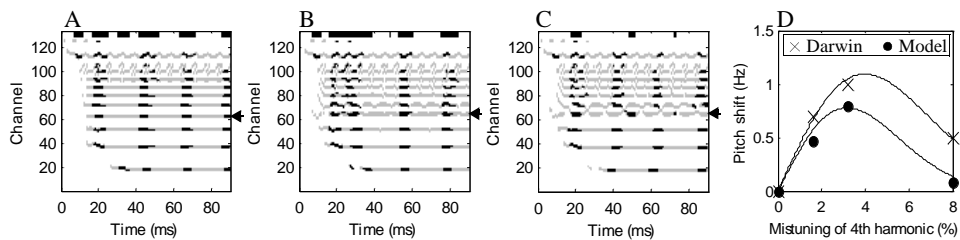

Figure 2: A,B,C: Network response to mistuning of the fourth harmonic of a 12 harmonic complex (0%, 6% and 8% respectively). ALI activity is shown at the top of each plot. Gray areas denote the presence of a segment and black areas denote oscillators in the active phase. Arrows show the focus of attentional interest. D: Pitch shift versus degree of mistuning. A Gaussian derivative is fitted to each data set.

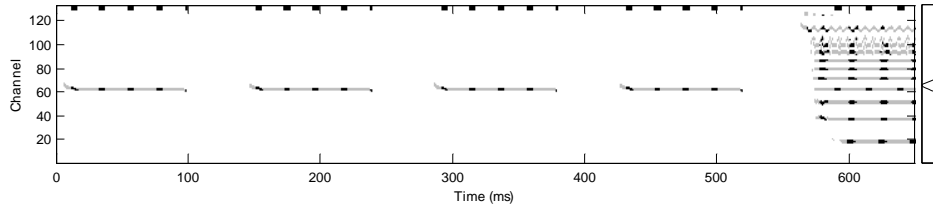

Figure 3: Captor tones preceding the complex capture the fourth harmonic into a separate stream. ALI activity (top) shows that this harmonic is the focus of attention and would be 'heard out'. The attentional interest vector ($A_k$) is shown to the right of the figure.

This behaviour is reproduced by our model (Figure 2). All the oscillators at frequency channels corresponding to harmonics are temporally synchronised for mistunings up to 8% (plots A and B) signifying that the harmonics belong to the same perceptual group. Mistunings beyond 8% cause the mistuned harmonic to become desynchronised from the rest of the complex (plot C) - two distinct perceptual groups are now present: one containing the fourth harmonic and the other containing the remainder of the complex tone. A comparison of the pitch shifts found by Darwin *et al.* and the shifts predicted by the model is shown in plot D. The pitch of the complex was calculated by creating a summary correlogram (similar to that used in section 2.2) using frequency channels contained within the complex tone group. Only segment channels below 1.1 kHz were used for this summary since low frequency (resolved) harmonics are known to dominate the pitch percept [8].

Darwin *et al.* also showed that the effect of mistuning was diminished when the fourth harmonic was 'captured' from the complex by four preceding tones at the same frequency. In this situation, no matter how small the mistuning, the harmonic is segregated from the complex and does not influence the pitch percept. Figure 3 shows the capture of the harmonic with no mistuning. Attentional interest is focused on the fourth harmonic: oscillator activity for the captor tone segments is synchronised with the ALI activity. During the 550 ms before the complex tone onset, the age tracker activities for the captor tone channels build up. When the complex tone begins, there is a significant age difference between the frequency channels stimulated by the fourth harmonic and those stimulated by the remainder of the complex. Such a difference prevents excitatory harmonicity connections from being made between the fourth harmonic and the remaining harmonics. This behaviour is consistent with the old-plus-new heuristic; a current acoustic event is interpreted as a continuation of a previous stimulus.

The old-plus-new heuristic can be further demonstrated by starting the fourth harmonic before the rest of the complex. Figure 4 shows the output of the model when the fourth harmonic is subject to a 50 ms onset asynchrony. During this time, the age trackers of channels excited by the fourth harmonic increase to a significantly higher value than those of the remaining harmonics. Once again, this prevents excitatory connections being made between the fourth harmonic and the other harmonically related segments. The early harmonic is desynchronised from the rest of the complex: two streams are formed. However, after a period of time, the importance of the onset asynchrony decreases as the channel ages approach their maximal values. Once this occurs, there is no longer any evidence to prevent excitatory links from being made between the fourth harmonic and the rest of the complex. Grouping by harmonicity then occurs for all segments: the complex and the early harmonic synchronise to form a single stream.

## 3.2 Auditory streaming

Within the framework presented here, auditory streaming is an emergent property; all events which occur over time, and are subject to attentional interest, are implicitly grouped. Two temporally separated events at different frequencies must both fall under the $A_k$ peak to be

grouped. It is the width of the $A_k$ peak that determines frequency separation-dependent streaming, rather than local connections between oscillators as in [10]. The build-up of streaming [1] is modelled by the leaky integrator in (9). Figure 5 shows the effect of two different frequency separations on the ability of the network to perform auditory streaming and shows a good match to experimental findings [1], [4]. At low frequency separations, both the high and low frequency segments fall under the attentional interest peak; this allows the oscillator activities of both frequency bands to influence the ALI and hence they are considered to be in the attentional foreground. At higher frequency separations, one of the frequency bands falls outside of the attentional peak (in this example, the high frequency tones fall outside) and hence it cannot influence the ALI. Such behaviour is not seen immediately, because the attentional interest vector is subject to a build up effect as described in (9). Initially the attentional interest is maximal across all frequencies; as the leaky integrator value increases, the interest peak begins to dominate and interest in other frequencies tends toward zero.

## 4 Discussion

A model of auditory attention has been presented which is based on previous neural oscillator work by Wang and colleagues [10], [11] but differs in two important respects. Firstly, our network is unidimensional; in contrast, Wang's approach employs a two-dimensional time-frequency grid for which there is weak physiological justification. Secondly, our model regards attention as a key factor in the stream segregation process. In our model, attentional interest may be consciously directed toward a particular stream, causing that stream to be selected as the attentional foreground.

Few auditory models have incorporated attentional effects in a plausible manner. For example, Wang's 'shifting synchronisation' theory [3] suggests that attention is directed towards a stream when its constituent oscillators reach the active phase. This contradicts experimental findings, which suggest that attention selects a single stream whose salience is increased for a sustained period of time [2]. Additionally, Wang's model fails to account for exogenous reorientation of attention to a sudden loud stimulus; the shifting synchronisation approach would multiplex it as normal with no attentional emphasis. By ensuring that the minimum $A_k$ value for the attentional interest is always non-zero, it is possible to weight activity outside of the attentional interest peak and force it to influence the ALI. Such weighting could be derived from a measure of the sound intensity present in each frequency channel.

We have demonstrated the model's ability to accurately simulate a number of perceptual phenomena. The time course of perception is well simulated, showing how factors such as mistuning and onset asynchrony can cause a harmonic to be segregated from a complex tone. It is interesting to note that a good match to Darwin's pitch shift data (Figure 2D) was only found when harmonically related segments below 1.1 kHz were used. The dominance of lower (resolved) harmonics on pitch is well known [8], and our findings suggest that the correlogram does not accurately model this aspect of pitch perception.

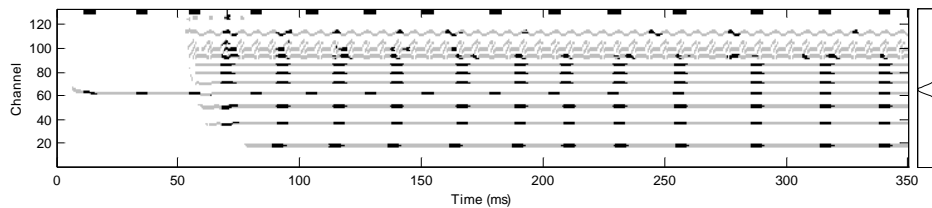

Figure 4: Asynchronous onset of the fourth harmonic causes it to segregate into a separate stream. The attentional interest vector ($A_k$) is shown to the right of the figure.

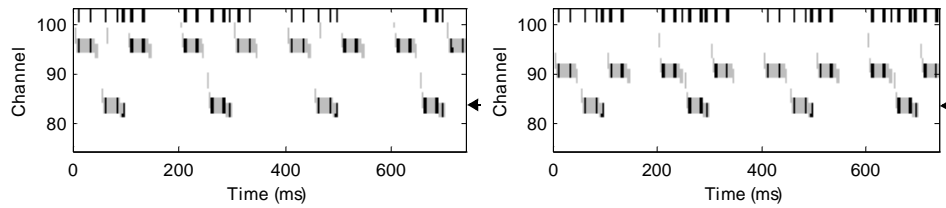

Figure 5: Auditory streaming at frequency separations of 5 semitones (left) and 3 semitones (right). Streaming occurs at the higher separation. The timescale of adaptation for the attentional interest has been reduced to aid the clarity of the figures.

The simulation of two tone streaming shows how the proposed attentional mechanism and its cross-frequency spread accounts for grouping of sequential events according to their proximity in frequency. A sequence of two tones will only stream if one set of tones fall outside of the peak of attentional interest. Frequency separations for streaming to occur in the model (greater than 3 to 4 semitones) are in agreement with experimental data, as is the timescale for the build-up of the streaming effect [1].

In summary, we have proposed a physiologically plausible model in which auditory streams are encoded by a unidimensional neural oscillator network. The network creates auditory streams according to grouping factors such as harmonicity, frequency proximity and common onset, and selects one stream as the attentional foreground. Current work is concentrating on expanding the system to include binaural effects, such as inter-ear attentional competition [4].

## References

[1] Anstis, S. & Saida, S. (1985) Adaptation to auditory streaming of frequency-modulated tones. *J. Exp. Psychol. Human* **11** 257-271.

[2] Bregman, A. S. (1990) *Auditory Scene Analysis*. Cambridge MA: MIT Press.

[3] Brown, G. J. & Cooke, M. (1994) Computational auditory scene analysis. *Comput. Speech Lang.* **8**, pp. 297-336.

[4] Carlyon, R. P., Cusack, R., Foxton, J. M. & Robertson, I. H. (2001) Effects of attention and unilateral neglect on auditory stream segregation. *J. Exp. Psychol. Human* **27**(1) 115-127.

[5] Darwin, C. J., Hukin, R. W. & Al-Khatib, B. Y. (1995) Grouping in pitch perception: Evidence for sequential constraints. *J. Acoust. Soc. Am.* **98**(2)Pt1 880-885.

[6] Joliot, M., Ribary, U. & Llinás, R. (1994) Human oscillatory brain activity near 40 Hz coexists with cognitive temporal binding. *Proc. Natl. Acad. Sci. USA* **91** 11748-51.

[7] Mondor, T. A. & Bregman, A. S. (1994) Allocating attention to frequency regions. *Percept. Psychophys.* **56**(3) 268-276.

[8] Moore, B. C. J. (1997) *An introduction to the psychology of hearing*. Academic Press.

[9] Spence, C. J., Driver, J. (1994) Covert spatial orienting in audition: exogenous and endogenous mechanisms. *J. Exp. Psychol. Human* **20**(3) 555-574.

[10] Wang, D. L. (1996) Primitive auditory segregation based on oscillatory correlation. *Cognitive Sci.* **20** 409-456.

[11] Wang, D. L. & Brown, G. J. (1999) Separation of speech from interfering sounds based on oscillatory correlation. *IEEE Trans. Neural Networks* **10** 684-697.
